# Spatial Latent Dirichlet Allocation

**Xiaogang Wang and Eric Grimson**

Computer Science and Artificial Intelligence Lab
Massachusetts Institute of Technology, Cambridge, MA, 02139, USA

xgwang@csail.mit.edu, welg@csail.mit.edu

## Abstract

In recent years, the language model Latent Dirichlet Allocation (LDA), which clusters co-occurring words into topics, has been widely applied in the computer vision field. However, many of these applications have difficulty with modeling the spatial and temporal structure among visual words, since LDA assumes that a document is a "bag-of-words". It is also critical to properly design "words" and "documents" when using a language model to solve vision problems. In this paper, we propose a topic model Spatial Latent Dirichlet Allocation (SLDA), which better encodes spatial structures among visual words that are essential for solving many vision problems. The spatial information is not encoded in the values of visual words but in the design of documents. Instead of knowing the partition of words into documents *a priori*, the word-document assignment becomes a random hidden variable in SLDA. There is a generative procedure, where knowledge of spatial structure can be flexibly added as a prior, grouping visual words which are close in space into the same document. We use SLDA to discover objects from a collection of images, and show it achieves better performance than LDA.

## 1 Introduction

Latent Dirichlet Allocation (LDA) [1] is a language model which clusters co-occurring words into topics. In recent years, LDA has been widely used to solve computer vision problems. For example, LDA was used to discover objects from a collection of images [2, 3, 4] and to classify images into different scene categories [5]. [6] employed LDA to classify human actions. In visual surveillance, LDA was used to model atomic activities and interactions in a crowded scene [7]. In these applications, LDA clustered low-level visual words (which were image patches, spatial and temporal interest points or moving pixels) into topics with semantic meanings (which corresponded to objects, parts of objects, human actions or atomic activities) utilizing their co-occurrence information.

Even with these promising achievements, however, directly borrowing a language model to solve vision problems has some difficulties. First, LDA assumes that a document is a bag of words, such that spatial and temporal structures among visual words, which are meaningless in a language model but important in many computer vision problems, are ignored. Second, users need to define the meaning of "documents" in vision problems. The design of documents often implies some assumptions on vision problems. For example, in order to cluster image patches, which are treated as words, into classes of objects, researchers treated images as documents [2]. This assumes that if two types of patches are from the same object class, they often appear in the same images. This assumption is reasonable, but not strong enough. As an example shown in Figure 1, even though sky is far from vehicles, if they often exist in the same images in some data set, they would be clustered into the same topic by LDA. Furthermore, since in this image most of the patches are sky and building, a patch on a vehicle is likely to be labeled as building or sky as well. These problems could be solved if the document of a patch, such as the yellow patch in Figure 1, only includes other

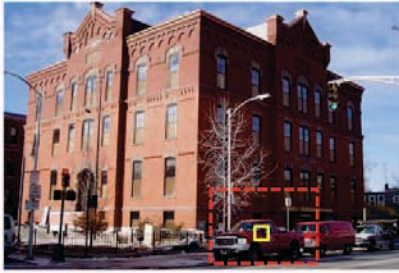

Figure 1: There will be some problems (see text) if the whole image is treated as one document when using LDA to discover classes of objects.

patches falling within its neighborhood, marked by the red dashed window in Figure 1, instead of the whole image. So a better assumption is that if two types of image patches are from the same object class, they are not only often in the same images but also close in space. We expect to utilize spatial information in a flexible way when designing documents for solving vision problems.

In this paper, we propose a Spatial Latent Dirichlet Allocation (SLDA) model which encodes the spatial structure among visual words. It clusters visual words (e.g. an eye patch and a nose patch), which often occur in the same images and are close in space, into one topic (e.g. face). This is a more proper assumption for solving many vision problems when images often contain several objects. It is also easy for SLDA to model activities and human actions by encoding temporal information. However the spatial or temporal information is not encoded in the values of visual words, but in the design of documents. LDA and its extensions, such as the author-topic model [8], the dynamic topic model [9], and the correlated topic model [10], all assume that the partition of words into documents is known *a priori*. A key difference of SLDA is that the word-document assignment becomes a hidden random variable. There is a generative procedure to assign words to documents. When visual words are close in space or time, they have a high probability to be grouped into the same document. Some approaches such as [11, 3, 12, 4] could also capture some spatial structures among visual words. [11] assumed that the spatial distribution of an object class could be modeled as Gaussian and the number of objects in the image was known. Both [3] and [4] first roughly segmented images using graph cuts and added spatial constraint using these segments. [12] modeled the spatial dependency among image patches as Markov random fields.

As an example application, we use the SLDA model to discover objects from a collection of images. As shown in Figure 2, there are different classes of objects, such as cows, cars, faces, grasses, sky, bicycles, etc., in the image set. And an image usually contains several objects of different classes. The goal is to segment objects from images, and at the same time, to label these segments as different object classes in an unsupervised way. It integrates object segmentation and recognition. In our approach images are divided into local patches. A local descriptor is computed for each image patch and quantized into a visual word. Using topic models, the visual words are clustered into topics which correspond to object classes. Thus an image patch can be labeled as one of the object classes. Our work is related to [2] which used LDA to cluster image patches. As shown in Figure 2, SLDA achieves much better performance than LDA. We will compare more results of LDA and SLDA in the experimental section.

## 2 Computation of Visual Words

To obtain the local descriptors, images are convolved with the filter bank proposed in [13], which is a combination of 3 Gaussians, 4 Laplacian of Gaussians, and 4 first order derivatives of Gaussians, and was shown to have good performance for object categorization. Instead of only computing visual words at interest points as in [2], we divide an image into local patches on a grid and densely sample a local descriptor for each patch. A codebook of size $W$ is created by clustering all the local descriptors in the image set using K-means. Each local patch is quantized into a visual word according to the codebook. In the next step, these visual words (image patches) will be further clustered into classes of objects. We will compare two clustering methods, LDA and SLDA.

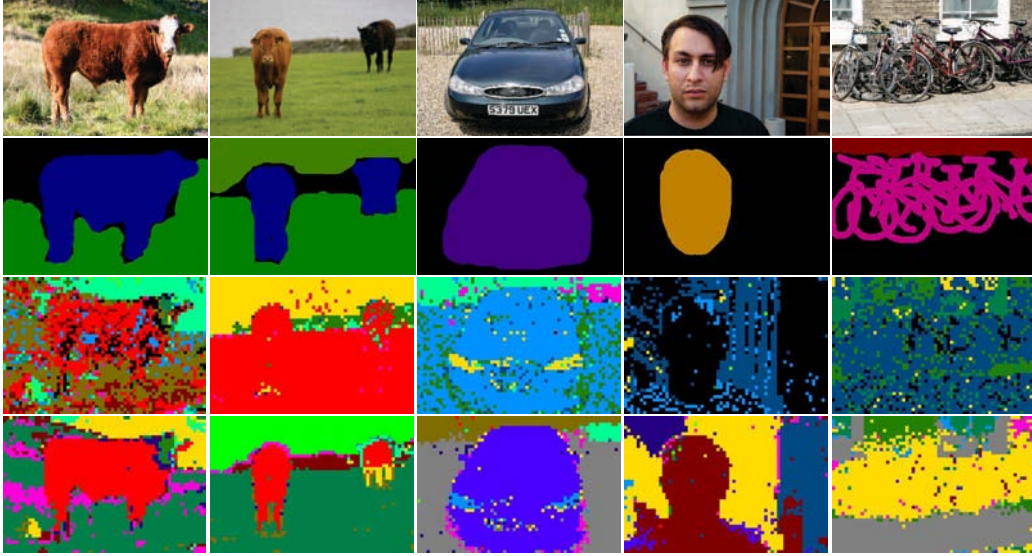

Figure 2: Given a collection of images as shown in the first row (which are selected from the MSRC image dataset [13]), the goal is to segment images into objects and cluster these objects into different classes. The second row uses manual segmentation and labeling as ground truth. The third row is the LDA result and the fourth row is the SLDA result. Under the same labeling approach, image patches marked in the same color are in one object cluster, but the meaning of colors changes across different labeling methods.

## 3 LDA

When LDA is used to solve our problem, we treat local patches of images as words and the whole image as a document. The graphical model of LDA is shown in Figure 3 (a). There are $M$ documents (images) in the corpus. Each document $j$ has $N_j$ words (image patches). $w_{ji}$ is the observed value of word $i$ in document $j$. All the words in the corpus will be clustered into $K$ topics (classes of objects). Each topic $k$ is modeled as a multinomial distribution over the codebook. and $\phi$ are Dirichlet prior hyperparameters. $_k$, $_j$, and $z_{ji}$ are hidden variables to be inferred. The generative process of LDA is:

1. For a topic $k$, a multinomial parameter $_k$ is sampled from Dirichlet prior $_k \quad Dir(\phi)$.
2. For a document $j$, a multinomial parameter $_j$ over the $K$ topics is sampled from Dirichlet prior $_j \quad Dir( )$.
3. For a word $i$ in document $j$, a topic label $z_{ji}$ is sampled from discrete distribution $z_{ji} \quad Discrete( _j)$.
4. The value $w_{ji}$ of word $i$ in document $j$ is sampled from the discrete distribution of topic $z_{ji}, w_{ji} \quad Discrete( _{z_{ji}})$.

$z_{ji}$ can be sampled through a Gibbs sampling procedure which integrates out $_j$ and $_k$ [14].

$$p(z_{ji} = k | \mathbf{z}_{\ ji} \mathbf{w} \quad \phi) \quad \frac{n^{(k)}_{\ ji\ w_{ji}} + \phi_{w_{ji}}}{\sum_{w=1}^{W} n^{(k)}_{\ ji\ w} + \phi_w} \quad \frac{n^{(j)}_{\ ji\ k} + _k}{\sum_{k=1}^{K} n^{(j)}_{\ ji\ k} + _k} \tag{1}$$

where $n^{(k)}_{\ ji\ w}$ is the number of words in the corpus with value $w$ assigned to topic $k$ excluding word $i$ in document $j$, and $n^{(j)}_{\ ji\ k}$ is the number of words in document $j$ assigned to topic $k$ excluding word $i$ in document $j$. Eq 1 is the product of two ratios: the probability of word $w_{ji}$ under topic $k$ and the probability of topic $k$ in document $j$. So LDA clusters the visual words often co-occurring in the same images into one object class.

As shown by some examples in Figure 2 (see more results in the experimental section), there are two problems in using LDA for object segmentation and recognition. The segmentation result is

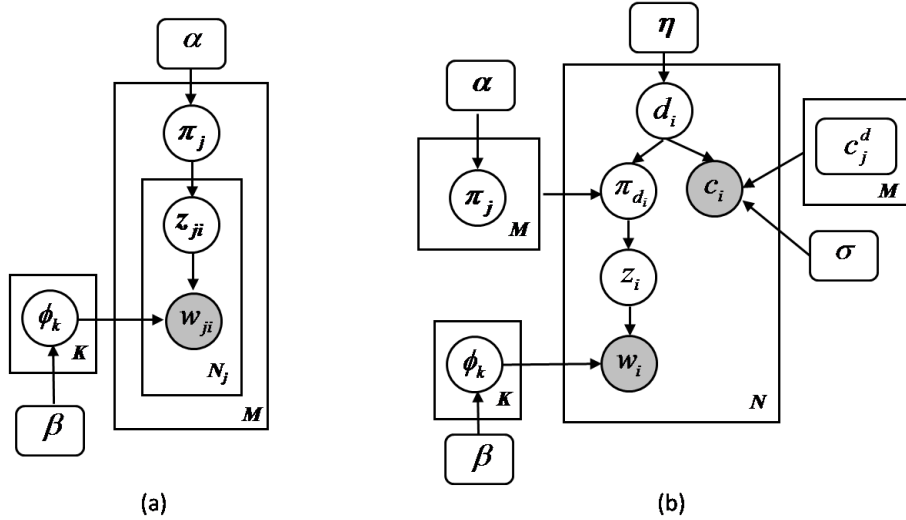

Figure 3: Graphical model of LDA (a) and SLDA (b). See text for details.

noisy since spatial information is not considered. Although LDA assumes that one image contains multiple topics, from experimental results we observe that the patches in the same image are likely to have the same labels. Since the whole image is treated as one document, if one object class, e.g. car in Figure 2, is dominant in the image, the second ratio in Eq 1 will lead to a large bias towards the car class, and thus the patches of street are also likely to be labeled as car. This problem could be solved if a local patch only considers its neighboring patches as being in the same document.

## 4 SLDA

We assume that if visual words are from the same class of objects, they not only often co-occur in the same images but also are close in space. So we try to group image patches which are close in space into the same documents. One straightforward way is to divide the image into regions as shown in Figure 4 (a). Each region is treated as a document instead of the whole image. However, since these regions are not overlapped, some patches, such as A (red patch) and B (cyan patch) in Figure 4 (a), even though very close in space, are assigned to different documents. In Figure 4 (a), patch A on the cow is likely to be labeled as grass, since most other patches in its document are grass. To solve this problem, we may put many overlapped regions, each of which is a document, on the images as shown in Figure 4 (b). If a patch is inside a region, it "could" belong to that document. Any two patches whose distance is smaller than the region size "could" belong to the same document if the regions are placed densely enough. We use the word "could" because each local patch is covered by several regions, so we have to decide to which document it belongs. Different from the LDA model, in which the word-document relationship is known *a priori*, we need a generative procedure assigning words to documents. If two patches are closer in space, they have a higher probability to be assigned to the same document since there are more regions covering both of them. Actually we can go even further. As shown in Figure 4 (c), each document can be represented by a point (marked by magenta circle) in the image, assuming its region covers the whole image. If an image patch is close to a document, it has a high probability to be assigned to that document.

The graphical model is shown in Figure 3 (b). In SLDA, there are $M$ documents and $N$ words in the corpus. A hidden variable $d_i$ indicates which document word $i$ is assigned to. For each document $j$ there is a hyperparameter $c_j^d = \left(g_j^d, x_j^d, y_j^d\right)$ known *a priori*. $g_j^d$ is the index of the image where document $j$ is placed and $\left(x_j^d, y_j^d\right)$ is the location of the document. For a word $i$, in addition to the observed word value $w_i$, its location $(x_i, y_i)$ and image index $g_i$ are also observed and stored in variable $c_i = (g_i, x_i, y_i)$. The generative procedure of SLDA is:

1. For a topic $k$, a multinomial parameter $\phi_k$ is sampled from Dirichlet prior $\phi_k \sim Dir(\beta)$.

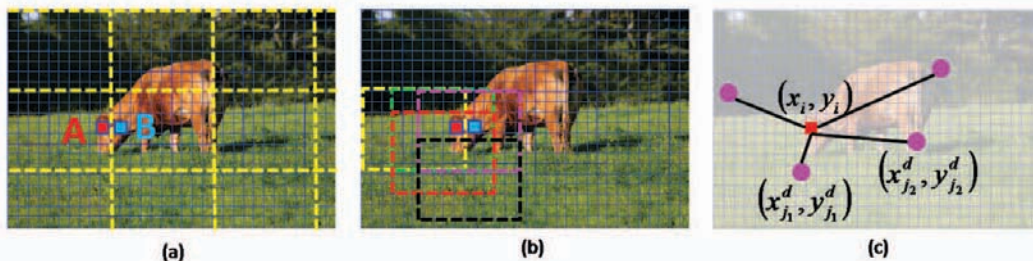

Figure 4: There are several ways to add spatial information among image patches when designing documents. (a): Divide the image into regions without overlapping. Each region, marked by a dashed window, corresponds to a document. Image patches inside the region are assigned to the corresponding document. (b): densely put overlapped regions over images. One image patch is covered by multiple regions. (c): Each document is associated with a point (marked in magenta color). These points are densely placed over the image. If a image patch is close to a document, it has a high probability to be assigned to that document.

2. For a document $j$, a multinomial parameter $\theta_j$ over the $K$ topics is sampled from Dirichlet prior $\theta_j \sim Dir(\alpha)$.

3. For a word (image patch) $i$, a random variable $d_i$ is sampled from prior $p(d_i|\sigma)$ indicating to which document word $i$ is assigned. We choose $p(d_i|\sigma)$ as a uniform prior.

4. The image index and location of word $i$ is sampled from distribution $p(c_i|c_{d_i}^d)$. We may choose this as a Gaussian kernel.

$$p((g_i, x_i, y_i)|g_{d_i}^d, x_{d_i}^d, y_{d_i}^d) \propto \pi_{g_{d_i}^d}(g_i) \exp\left(-\frac{(x_{d_i}^d - x_i)^2 + (y_{d_i}^d - y_i)^2}{2\gamma}\right)$$

$p(c_i|c_{d_i}^d) = 0$ if the word and the document are not in the same image.

5. The topic label $z_i$ of word $i$ is sampled from the discrete distribution of document $d_i$, $z_i \sim Discrete(\theta_{d_i})$.

6. The value $w_i$ of word $i$ is sampled from the discrete distribution of topic $z_i$, $w_i \sim Discrete(\phi_{z_i})$.

## 4.1 Gibbs Sampling

$z_i$ and $d_i$ can be sampled through a Gibbs sampling procedure integrating out $\phi_k$ and $\theta_j$. In SLDA the conditional distribution of $z_i$ given $d_i$ is the same as in LDA.

$$p(z_i = k|d_i = j, \mathbf{d}_{-i}, \mathbf{z}_{-i}, \mathbf{w}, \alpha, \phi) \propto \frac{n_{-i,w_i}^{(k)} + \phi_{w_i}}{\sum_{w=1}^{W} n_{-i,w}^{(k)} + \phi_w} \cdot \frac{n_{-i,k}^{(j)} + \alpha_k}{\sum_{k'=1}^{K} n_{-i,k'}^{(j)} + \alpha_{k'}} \qquad (2)$$

where $n_{-i,w}^{(k)}$ is the number of words in the corpus with value $w$ assigned to topic $k$ excluding word $i$, and $n_{-i,k}^{(j)}$ is the number of words in document $j$ assigned to topic $k$ excluding word $i$. This is easy to understand since if the word-document assignment is fixed, SLDA is the same as LDA.

In addition, we also need to sample $d_i$ from the conditional distribution given $z_i$.

$$p\left(d_i = j|z_i = k, \mathbf{z}_{-i}, \mathbf{d}_{-i}, c_i, c_j^d, \alpha, \phi, \sigma\right)$$
$$\propto p(d_i = j|\sigma)\, p\left(c_i|c_j^d\right)\, p\left(z_i = k|\mathbf{z}_{-i}, d_i = j, \mathbf{d}_{-i}\right)$$

$p\left(z_i = k|\mathbf{z}_{-i}, d_i = j, \mathbf{d}_{-i}\right)$ is obtained by integrating out $\theta_j$.

$$p\left(z_i = k|\mathbf{z}_{-i}, d_i = j, \mathbf{d}_{-i}\right) = \prod_{j=1}^{M} \int p(\theta_j|\alpha)\, p(\mathbf{z}_j|\theta_{ji})\, d\theta_j$$

$$= \prod_{j=1}^{M} \frac{\Gamma\left(\sum_{k=1}^{K}\alpha_k\right)}{\prod_{k=1}^{K}\Gamma(\alpha_k)} \cdot \frac{\prod_{k=1}^{K} \Gamma\left(n_k^{(j)} + \alpha_k\right)}{\Gamma\left(\sum_{k=1}^{K} n_k^{(j)} + \sum_{k=1}^{K}\alpha_k\right)}$$

We choose $p\left(d_i = j|\eta\right)$ as a uniform prior and $p\left(c_i|c_j^d, \sigma\right)$ as a Gaussian kernel. Thus the conditional distribution of $d_i$ is

$$p\left(d_i = j|z_i = k, \mathbf{z}_{-i}, \mathbf{d}_{-i}, c_i, \{c_{j'}^d\}, \alpha, \beta, \eta, \sigma\right)$$

$$\propto \quad \delta_{g_j^d}\left(g_i\right) \cdot e^{-\frac{\left(x_j^d - x_i\right)^2 + \left(y_j^d - y_i\right)^2}{\sigma^2}} \cdot \frac{n_{-i,k}^{(j)} + \alpha_k}{\sum_{k'=1}^{K}\left(n_{-i,k'}^{(j)} + \alpha_{k'}\right)} \quad (3)$$

Word $i$ is likely to be assigned to document $j$ if they are in the same image, close in space and word $i$ has the same topic label as other words in document $j$. In real applications, we only care about the distribution of $z_i$ while $d_j$ can be marginalized by simply ignoring its samples. From Eq 2 and 3, we observed that a word tends to have the same topic label as other words in its document and words closer in space are more likely to be assigned to the same documents. So essentially under SLDA a word tends to be labeled as the same topic as other words close to it. This satisfies our assumption that visual words from the same object class are closer in space.

Since we densely place many documents over one image, during Gibbs sampling some documents are only assigned a few words and the distributions cannot be well estimated. To solve this problem we replicate each image patch to get many particles. These particles have the same word value and location but can be assigned to different documents and have different labels. Thus each document will have enough samples of words to estimate the distributions.

### 4.2 Discussion

SLDA is a flexible model intended to encode spatial structure among image patches and design documents. If there is only one document placed over one image, SLDA simply reduces to LDA. If $p(c_i|c_j^d)$ is an uniform distribution inside a local region, SLDA implements the scheme described in Figure 4 (b). If these local regions are not overlapped, it is the case of Figure 4 (a). There are also other possible ways to add spatial information by choosing different spatial priors $p(c_i|c_j^d)$. In SLDA, the spatial information is used when designing documents. However the object class model $\phi_k$, simply a multinomial distribution over the codebook, has no spatial structure. So the objects of a class could be in any shape and anywhere in the images, as long as they smoothly distribute in space. By simply adding a time stamp to $c_i$ and $c_j^d$, it is easy for SLDA to encode temporal structure among visual words. So SLDA also can be applied to human action and activity analysis.

## 5   Experiments

We test LDA and SLDA on the MSRC image dataset [13] with 240 images. Our codebook size is 200 and the topic number is 15. In Figure 2, we show some examples of results using LDA and SLDA. Colors are used indicate different topics. The results of LDA are noisy and within one image most of the patches are labeled as one topic. SLDA achieves much better results than LDA. The results are smoother and objects are well segmented. The detection rate and false alarm rate of four classes, cows, cars, faces, and bicycles are shown in Table 1. They are counted in pixels. We use the manual segmentation and labeling in [13] as ground truth.

The two models are also tested on a tiger video sequence with 252 frames. We treat all the frames in the sequence as an image collection and ignore their temporal order. Figure 5 shows their results on two sampled frames. Please see the result of the whole video sequence from our website [15]. Using LDA, usually there are one or two dominant topics distributed like noise in a frame. Topics change as the video background changes. LDA cannot segment out any objects. SLDA clusters image patches into tigers, rock, water, and grass. If we choose the topic of tiger, as shown in the last row of Figure 5, all the tigers in the video can be segmented out.

## 6   Conclusion

We propose a novel Spatial Latent Dirichlet Allocation model which clusters co-occurring and spatially neighboring visual words into the same topic. Instead of knowing word-document assignment *a priori*, SLDA has a generative procedure partitioning visual words which are close in space into the same documents. It is also easy to extend SLDA to including temporal information.

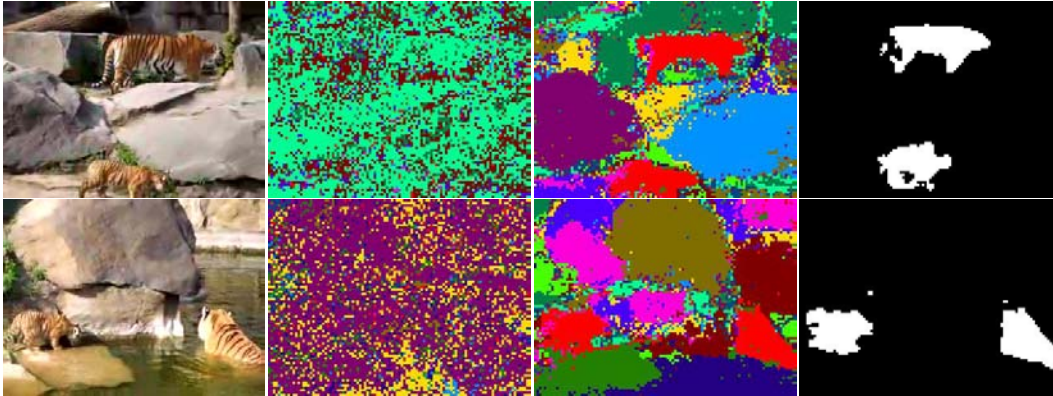

Figure 5: Discovering objects from a video sequence. The first column shows two frames in the video sequence. In the second column, we label the patches in the two frames as different topics using LDA. The thrid column plots the topic labels using SLDA. The red color indicates the topic of tigers. In the fourth column, we segment tigers out by choosing the topic marked in red.

Table 1: Detection(D) rate and False Alarm (FA) rate of LDA and SLDA on the MSRC data set

|          | cows   | cars   | faces  | bicycles |
|----------|--------|--------|--------|----------|
| LDA(D)   | 0.3755 | 0.5552 | 0.7172 | 0.5563   |
| SLDA(D)  | 0.5662 | 0.6838 | 0.6973 | 0.5661   |
| LDA(FA)  | 0.5576 | 0.3963 | 0.5862 | 0.5285   |
| SLDA(FA) | 0.0334 | 0.2437 | 0.3714 | 0.4217   |

# 7 Acknowledgement

The authors wish to acknowledge DSO National Laboratory of Singapore for partially supporting this research.

# References

[1] D. M. Blei, A. Y. Ng, and M. I. Jordan. Latent dirichlet allocation. *Journal of Machine Learning Research*, 3:993–1022, 2003.

[2] J. Sivic, B. C. Russell, A. A. Efros, A. Zisserman, and W. T. Freeman. Discovering object categories in image collections. In *Proc. ICCV*, 2005.

[3] B. C. Russell, A. A. Efros, J. Sivic, W. T. Freeman, and A. Zisserman. Using multiple segmentations to discover objects and their extent in image collections. In *Proc. CVPR*, 2006.

[4] L. Cao and L. Fei-Fei. Spatially coherent latent topic model for concurrent object segmentation and classification. In *Proc. ICCV*, 2007.

[5] L. Fei-Fei and P. Perona. A bayesian hierarchical model for learning natural scene categories. In *Proc. CVPR*, 2005.

[6] J. C. Niebles, H. Wang, and L. Fei-Fei. Unsupervised learning of human action categories using spatial-temporal words. In *Proc. BMVC*, 2006.

[7] X. Wang, X. Ma, and E. Grimson. Unsupervised activity perception by hierarchical bayesian models. In *Proc. CVPR*, 2007.

[8] M. Rosen-Zvi, T. Griffiths, M. Steyvers, and P. Smyth. The author-topic model for authors and documents. In *Proc. of Uncertainty in Artificial Intelligence*, 2004.

[9] D. Blei and J. Lafferty. Dynamic topic models. In *Proc. ICML*, 2006.

[10] D. Blei and J. Lafferty. Correlated topic models. In *Proc. NIPS*, 2006.

[11] E. B. Sudderth, A. Torralba, W. T. Freeman, and A. S. Willsky. Learning hierarchical models of scenes, objects, and parts. In *Proc. ICCV*, 2005.

[12] J. Verbeek and B. Triggs. Region classification with markov field aspect models. In *Proc. CVPR*, 2007.

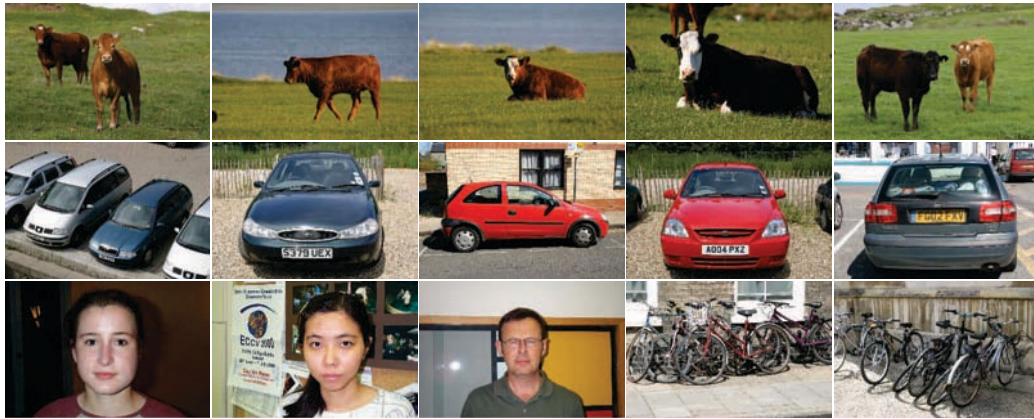

(a)

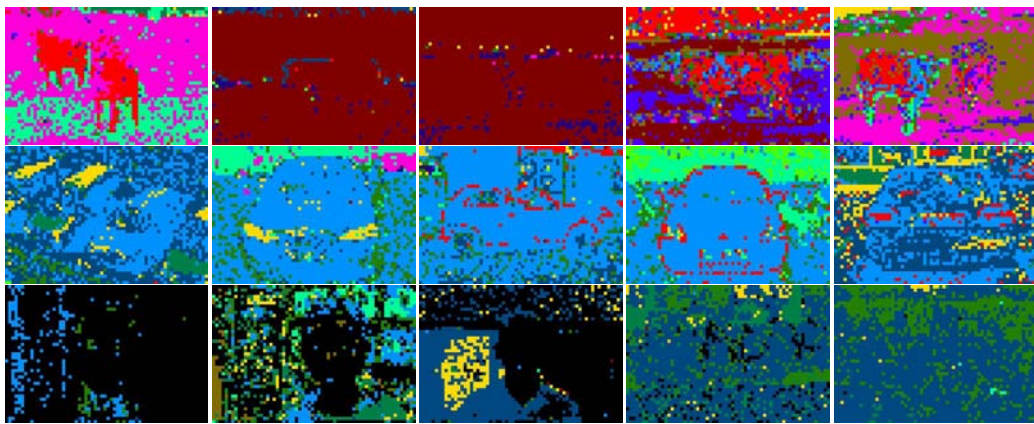

(b)

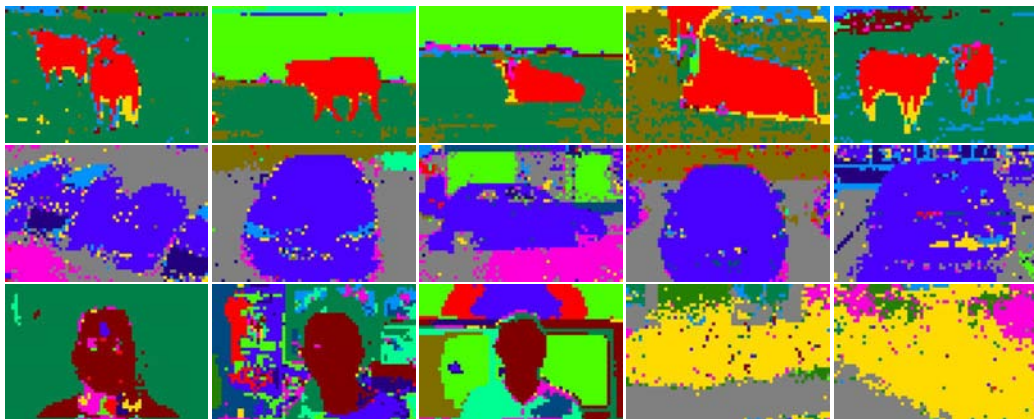

(c)

Figure 6: Examples of experimental results on the MSRC image data set. (a): original images; (b): LDA results; (c) SLDA results.

[13] J. Winn, A. Criminisi, and T. Minka. Object categorization by learned universal visual dictionary. In *Proc. ICCV*, 2005.

[14] T. Griffiths and M. Steyvers. Finding scientific topics. In *Proc. of the National Academy of Sciences*, 2004.

[15] http://people.csail.mit.edu/xgwang/slda.html.

